# Associative Memory in a Simple Model of Oscillating Cortex

**Bill Baird**
Dept Molecular and Cell Biology,
U.C.Berkeley, Berkeley, Ca. 94720

## ABSTRACT

A generic model of oscillating cortex, which assumes "minimal" coupling justified by known anatomy, is shown to function as an associative memory, using previously developed theory. The network has explicit excitatory neurons with local inhibitory interneuron feedback that forms a set of nonlinear oscillators coupled only by long range excitatory connections. Using a local Hebb-like learning rule for primary and higher order synapses at the ends of the long range connections, the system learns to store the kinds of oscillation amplitude patterns observed in olfactory and visual cortex. This rule is derived from a more general "projection algorithm" for recurrent analog networks, that analytically guarantees content addressable memory storage of continuous periodic sequences — capacity: N/2 Fourier components for an N node network — no "spurious" attractors.

## 1   Introduction

This is a sketch of recent results stemming from work which is discussed completely in [1, 2, 3]. Patterns of 40 to 80 hz oscillation have been observed in the large scale activity of olfactory cortex [4] and visual neocortex [5], and shown to predict the olfactory and visual pattern recognition responses of a trained animal. It thus appears that cortical computation in general may occur by dynamical interaction of resonant modes, as has been thought to be the case in the olfactory system. Given the sensitivity of neurons to the location and arrival times of dendritic input, the

sucessive volleys of pulses that are generated by the collective oscillation of a neural net may be ideal for the formation and reliable longe range transmission of the collective activity of one cortical area to another. The oscillation can serve a macroscopic clocking function and entrain the relevant microscopic activity of disparate cortical regions into well defined phase coherent macroscopic collective states which overide uncorrelated microscopic activity. If this view is correct, then oscillatory network modules form the actual cortical substrate of the diverse sensory, motor, and cognitive operations now studied in static networks, and it must ultimately be shown how those functions can be accomplished with these dynamic networks.

In particular, we are interested here in modeling category learning and object recognition, *after* feature preprocessing. Equivalence classes of ratios of feature outputs in feature space must be established as prototype "objects" or categories that are invariant over endless sensory instances. Without categories, the world never repeats. This is the kind of function generally hypothesized for prepyriform cortex in the olfactory system[6], or inferotemporal cortex in the visual system. It is a different oscillatory network function from the feature "binding", or clustering role that is hypothesized for "phase labels" in primary visual cortex [5], or from the "decision states" hypothesized for the olfactory bulb by Li and Hopfield. In these preprocessing systems, there is no modification of connections, and no learning of particular perceptual objects. For category learning, full adaptive cross coupling is required so that all possible input feature vectors may be potential attractors. This is the kind of anatomical structure that characterizes prepyriform and inferotemporal cortex. The columns there are less structured, and the associational fiber system is more prominent than in primary cortex. Man shares this same high level "association" cortex structure with cats and rats. Phylogenetically, it is the preprocessing structures of primary cortex that have grown and evolved to give us our expanded capabilities. While the bulk of our pattern recognition power may be contributed by the clever feature preprocessing that has developed, the object classification system seems the most likely locus of the learning changes that underlie our daily conceptual evolution. That is the phenomenon of ultimate interest in this work.

## 2    Minimal Model of Oscillating Cortex

Analog state variables, recurrence, oscillation, and bifurcation are hypothesized to be essential features of cortical networks which we explore in this approach. Explicit modeling of known excitatory and inhibitory neurons, and use of only known long range connections is also a basic requirement to have a biologically feasible network architecture. We analyse a "minimal" model that is intended to assume the least coupling that is justified by known anatomy, and use simulations and analytic results proved in [1, 2] to argue that an oscillatory associative memory function can be realized in such a system. The network is meant only as a cartoon of the real biology, which is designed to reveal the general mathematical principles and mechanisms by which the actual system might function. Such principles can then be observed or applied in other contexts as well.

Long range excitatory to excitatory connections are well known as "associational" connections in olfactory cortex[6], and cortico-cortico connections in neocortex. Since our units are neural populations, we know that some density of full cross-coupling exists in the system[6], and our weights are the average synaptic strengths of these connections. There is little problem at the population level with coupling symmetry in these average connection strenghts emerging from the operation of an outer product learning rule on initially random connections. When the network units are neuron pools, analog state variables arise naturally as continuous local pulse densities and cell voltage averages. Smooth sigmoidal population input-output functions, whose slope increases with arousal of the animal, have been measured in the olfactory system[4]. Local inhibitory "interneurons" are a ubiquitous feature of the anatomy of cortex throughout the brain[5]. It is unlikely that they make long range connections (> 1 mm) by themselves. These connections, and even the debated interconnections between them, are therefore left out of a minimal model. The resulting network is actually a fair caricature of the well studied circuitry of olfactory (prepyriform) cortex. This is thought to be one of the clearest cases of a real biological network with associative memory function [6]. Although neocortex is far more complicated, it may roughly be viewed as two olfactory cortices stacked on top of each other. We expect that analysis of this system will lend insight into mechanisms of associative memory there as well. In [3] we show that this model is capable of storing complicated multifrequency spatio-temporal trajectories, and argue that it may serve as a model of memory for sequences of actions in motor cortex.

For an $N$ dimensional system, the "minimal" coupling structure is described mathematically by the matrix

$$T = \begin{bmatrix} W & -hI \\ gI & 0 \end{bmatrix},$$

where $W$ is the $N/2 \times N/2$ matrix of excitatory interconnections, and $gI$ and $hI$ are $N/2 \times N/2$ identity matrices multiplied by the positive scalars $g$, and $h$. These give the strength of coupling around local inhibitory feedback loops. A state vector is composed of local average cell voltages for $N/2$ excitatory neuron populations $\vec{x}$ and $N/2$ inhibitory neuron populations $\vec{y}$ (hereafter notated as $x, y \in \mathbf{R}^{N/2}$). Standard network equations with this coupling might be, in component form,

$$\dot{x}_i = -\tau x_i - h\sigma(y_i) + \sum_{j=1}^{N/2} W_{ij}\sigma(x_j) + b_i \tag{1}$$

$$\dot{y}_i = -\tau y_i + g\sigma(x_i), \tag{2}$$

where $\sigma(x) = tanh(x)$ or some other sigmoidal function symmetric about 0. Intuitively, since the inhibitory units $y_i$ receive no direct input and give no direct output, they act as hidden units that create oscillation for the amplitude patterns stored in the excitatory cross-connections $W$. This may be viewed as a simple generalization of the analog "Hopfield" network architecture to store periodic instead of static attractors.

If we expand this network to third order in a Taylors series about the origin, we get a network that looks something like,

$$\dot{x}_i \;=\; -\tau x_i - h y_i + \sum_{j=1}^{N/2} W_{ij} x_j - \sum_{jkl=1}^{N/2} W_{ijkl} x_j x_k x_l + b_i, \tag{3}$$

$$\dot{y}_i \;=\; -\tau y_i + g x_i, \tag{4}$$

where $\sigma'(0) = 1$, and $\frac{1}{3!}\sigma'''(0)(<0)$ is absorbed into $W_{ijkl}$. A sigmoid symmetric about zero has odd symmetry, and the even order terms of the expansion vanish, leaving the cubic terms as the only nonlinearity. The actual expansion of the excitatory sigmoids in (1,2) (in this coordinate system) will only give cubic terms of the form $\sum_{j=1}^{N/2} W_{ij} x_j^3$. The competitive (negative) cubic terms of (3) therefore constitute a more general and directly programmable nonlinearity that is independent of the linear terms. They serve to create multiple periodic attractors by causing the oscillatory modes of the linear term to compete, much as the sigmoidal nonlinearity does for static modes in a Hopfield network. Intuitively, these terms may be thought of as sculpting the maxima of a "saturation" landscape into which the stored linear modes with positive eigenvalues expand, and positioning them to lie in the directions specified by the eigenvectors of these modes to make them stable. A precise definition of this landscape is given by a strict Liapunov function in a special polar coordinate system[1, 3]. Since we have had no success storing *multiple* oscillatory attractors in the sigmoid net (1,2) by any learning rule, we are driven to take this very effective higher order net seriously as a biological model. From a physiological point of view, (3,4) may be considered a model of a biological network which is operating in the linear region of the known *axonal* sigmoid nonlinearities[4], and contains instead sigma-pi units or higher order *synaptic* nonlinearities.

## 2.1 Biological justification of the higher order synapses

Using the long range excitatory connections available, the higher order synaptic weights $W_{ijkl}$ can conceivably be realized locally in the axo-dendritic interconnection plexus known as "neuropil". This a feltwork of tiny fibers so dense that it's exact circuitry is impossible to investigate with present experimental techniques. Single axons are known to bifurcate into multiple branches that contribute separate synapses to the dendrites of target cells. It is also well known that neighboring synapses on a dendrite can interact in a nonlinear fashion that has been modeled as higher order synaptic terms by some researchers. It has been suggested that the neuropil may be dense enough to allow the crossing of every possible combination of $jkl$ axons in the vicinity of some dendritic branch of at least one neuron in neuron pool $i$ (B. Mel). Trophic factors stimulated by the coactivation of the axons and the dendrite could cause these axons to form of a "cluster" of nearby synapses on the dendrite to realize a $jkl$ product synapse. The required higher order terms could thus be created by a Hebb-like process. The use of competitive cubic cross terms may therefore be viewed physiologically as the use of this complicated nonlinear *synaptic/dendritic* processing, as the decision making nonlinearity in the system, as

opposed to the usual sigmoidal *axonal* nonlinearity. There are more weights in the cubic synaptic terms, and the network nonlinearity can be programmed in detail.

## 3    Analysis

The real eigenvectors of $W$ give the magnitudes of the complex eigenvectors of $T$.

**Theorem 3.1** *If $\alpha$ is a real eigenvalue of the $N/2 \times N/2$ matrix $W$, with corresponding eigenvector $x$, then the $N \times N$ matrix*

$$T = \begin{bmatrix} W & -hI \\ gI & 0 \end{bmatrix}$$

*has a pair of complex conjugate eigenvalues $\lambda_{1,2} = 1/2(\alpha \pm \sqrt{\alpha^2 - 4hg}) = 1/2(\alpha \pm i\omega)$, for $\alpha^2 < 4hg$ , where $\omega = \sqrt{4hg - \alpha^2}$. The corresponding complex conjugate pair of eigenvectors are*

$$\begin{bmatrix} x \\ \frac{\alpha+\omega}{2h}x \end{bmatrix} \pm i \begin{bmatrix} x \\ \frac{\alpha-\omega}{2h}x \end{bmatrix} .$$

The proof of this theorem is given in [2]. To more clearly see the amplitude and phase patterns, we can convert to a magnitude and phase representation, $z = |z|e^{i\theta}$, where $|z_i| = \sqrt{\Re_{z_i}^2 + \Im_{z_i}^2}$, and $\theta_i = \arctan(\Im_{z_i})/(\Re_{z_i})$. We get, $|z_{x_i}| = \sqrt{x_i^2 + x_i^2} = \sqrt{2}|x_i|$ , and

$$|z_{y_i}| = \sqrt{\frac{2(\alpha^2 + \omega^2)}{4h^2}x_i^2} = \sqrt{\frac{4hg}{2h^2}}|x_i| = \sqrt{\frac{2g}{h}}|x_i|.$$

Now $\theta_x = \arctan 1 = \pi/4, \theta_y = \arctan\frac{\alpha-\omega}{\alpha+\omega}$. Dividing out the common $\sqrt{2}$ factor in the magnitudes, we get eigenvectors that clearly display the amplitude patterns of interest.

$$\begin{bmatrix} |x|e^{i\theta_x} \\ \sqrt{\frac{g}{h}}|x|e^{i\theta_y} \end{bmatrix} , \quad or \quad \begin{bmatrix} |x|\cos\theta_x \\ \sqrt{\frac{g}{h}}|x|\cos\theta_y \end{bmatrix} \pm i \begin{bmatrix} |x|\sin\theta_x \\ \sqrt{\frac{g}{h}}|x|\sin\theta_y \end{bmatrix}$$

Because of the restricted coupling, the oscillations possible in this network are standing waves, since the phase $\theta_x, \theta_y$ is constant for each kind of neuron $x$ and $y$, and differs only between them. This is basically what is observed in the olfactory bulb (primary olfactory cortex) and prepyriform cortex. The phase of inhibitory components $\theta_y$ in the bulb lags the phase of the excitatory components $\theta_x$ by approximately 90 degrees. It is easy to choose $\alpha$ and $\omega$ in this model to get phase lags of nearly 90 degrees.

### 3.1    Learning by the projection algorithm

From the theory detailed in [1], we can program any linearly independent set of eigenvalues and eigenvectors into $W$ by the "projection" operation $W = BDB^{-1}$, where $B$ has the desired eigenvectors as columns, and $D$ is a diagonal matrix of the desired eigenvalues. Because the complex eigenvectors of $T$ follow from these

learned for $W$, we can form a projection matrix $P$ with those eigenvectors of $T$ as columns. Forming also a matrix $J$ of the complex eigenvalues of T in blocks along the diagonal, we can project directly to get T. If general cubic terms $T_{ijkl}\, x_j x_k x_l$, also given by a specific projection operation, are added to network equations with linear terms $T_{ij}\, x_j$, the complex modes (eigenvectors) of the linearization are analytically guaranteed by the projection theorem[1] to characterize the periodic attractors of the network vector field. Chosen "normal form" coeficients $Amn$ [1] are projected to get the higher order synaptic weights $T_{ijkl}$ for these general cubic terms. Together, these operations constitute the "normal form projection algorithm":

$$T = PJP^{-1} \ , \quad T_{ijkl} = \sum_{m,n=1}^{N} P_{im} A_{mn} P_{mj}^{-1} P_{nk}^{-1} P_{nl}^{-1} .$$

Either member of the pair of complex eigenvectors shown above will suffice as the eigenvector that is entered in the $P$ matrix for the projection operation. For real and imaginary component columns in P,

$$P = \begin{bmatrix} |x^s|\cos\theta_x^s & |x^s|\sin\theta_x^s & \cdots \\ \sqrt{\frac{q}{h}}|x^s|\cos\theta_y^s & \sqrt{\frac{q}{h}}|x^s|\sin\theta_y^s & \cdots \end{bmatrix} \Rightarrow X^s(t) = \begin{bmatrix} |x^s|e^{i\theta_x^s + i\omega^s t} \\ \sqrt{\frac{q}{h}}|x^s|e^{i\theta_y^s + i\omega^s t} \end{bmatrix},$$

where $X^s(t)$ is an expression for the periodic attractor established for pattern $s$ when this $P$ matrix is used in the projection algorithm.

The general cubic terms $T_{ijkl}\, x_j x_k x_l$, however, require use of unlikely long range inhibitory connections. Simulations of two and four oscillator networks thus far (N=4 and N=8), reveal that use of the higher order terms for only the anatomically justified long range excitatory connections $W_{ijkl}$, as in the cubic net (3,4), is effective in storing randomly chosen sets of desired patterns. The behavior of this network is very close to the theoretical ideal guaranteed above for a network with general higher order terms. There is no alteration of stored oscillatory patterns when the reduced coupling is used.

We have at least general analytic justification for this. "Normal form" theory[1, 3] guarantees that many other choices of weights will do the same job as the those found by the projection operation, but does not in general say how to find them. Latest work shows that a perturbation theory calculation of the normal form coefficients for general high dimensional cubic nets is tractable and in principle permits the removal of all but $N^2$ of the $N^4$ higher order weights normally produced by the projection algorithm. We have already incorporated this in an improved learning rule (non-Hebbian thus far) which requires even fewer of the excitatory higher order weights $((N)^2$ instead of the $(N/2)^4$ used in (3)), and are exploring the size of the "neighborhood" of state space about the origin in which the rule is effective. This should lead as well to a rigorous proof of the performance of these networks.

## 3.2   Learning by local Hebb rules

We show further in [2, 1] that for orthonormal static patterns $x^s$, the projection operation for the $W$ matrix reduces to an outer product, or "Hebb" rule, and the

projection for the higher order weights becomes a *multiple* outer product rule:

$$W_{ij} = \sum_{s=1}^{N/2} \alpha^s x_i^s x_j^s \ , \qquad W_{ijkl} = c\, \delta_{ij}\delta_{kl} - d\sum_{s=1}^{N/2} x_i^s x_j^s x_k^s x_l^s \ . \qquad (5)$$

The first rule is guaranteed to establish desired patterns $x^s$ as eigenvectors of the matrix $W$ with corresponding eigenvalues $\alpha^s$. The second rule, with $c > d$, gives higher order weights for the cubic terms in (3) that ensure the patterns defined by these eigenvectors will appear as attractors in the network vectorfield. The outer product is a *local* synapse rule for synapse $ij$, that allows additive and incremental learning. The system can be truly self-organizing because the net can modify itself based on its own activity. The rank of the coupling matrix $W$ and $T$ grows as more memories are learned by the Hebb rule, and the unused capacity appears as a degenerate subspace with all zero eigenvalues. The flow is thus directed toward regions of the state space where patterns are stored.

In the minimal net, real eigenvectors learned for $W$ are converted by the network structure to standing wave oscillations (constant phase) with the absolute value of those eigenvectors as amplitudes. From the mathematical perspective, there are $(N/2)!$ eigenvectors with different permutations of the signs of the same components, which lead to the same positive amplitude vector. This means that nonorthogonal amplitude patterns may be stored by the Hebb rule on the excitatory connections, since there may be many ways to find a perfectly orthonormal set of eigenvectors for $W$ that stores a given set of nonorthogonal amplitude vectors. Given the complexity of dendritic processing discussed previously, it is not impossible that there is some distribution of the signs of the final effect of synapses from excitatory neurons that would allow a biological system to make use of this mathematical degree of freedom.

For different input objects, feature preprocessing in primary and secondary sensory cortex may be expected to orthogonalize outputs to the object recognition systems modeled here. When the rules above are used for nonorthogonal patterns, the eigenvectors of $W$ and $T$ are no longer given directly by the Hebb rule, and we expect that the kind of performance found in Hopfield networks for nonorthogonal memories will obtain, with reduced capacity and automatic clustering of similar exemplars. Investigation of this unsupervised induction of categories from training examples will be the subject of future work[3].

### 3.3   Architectural Variations — Olfactory Bulb Model

Another biologically interesting architecture which can store these kinds of patterns is one with associational excitatory to inhibitory cross-coupling. This may be a more plausible model of the olfactory bulb (primary olfactory cortex) than the one above. Experimental work of Freeman suggests an associative memory function for this cortex as well[4]. The evidence for long range excitatory to excitatory coupling in the olfactory bulb is much weaker than that for the prepyriform cortex. Long range excitatory tracts connecting even the two halves of the bulb are known, but anatomical data thus far show these axons entering only the inhibitory granuel cell

layers.

$$T = \begin{bmatrix} gI & -hI \\ W & 0 \end{bmatrix} \quad, \quad \lambda_{1,2} = 1/2(g \pm \sqrt{g^2 - 4\alpha g}) = 1/2(g \pm i\omega),$$

for $g^2 < 4\alpha g$, where $\omega = \sqrt{4\alpha g - g^2}$. The eigenvectors are,

$$\begin{bmatrix} x \\ \frac{g+\omega}{2h}x \end{bmatrix} \pm i \begin{bmatrix} x \\ \frac{g-\omega}{2h}x \end{bmatrix}, \quad \Rightarrow \quad P = \begin{bmatrix} |x^s|\cos\theta_x^s & |x^s|\sin\theta_x^s & \cdots \\ \sqrt{\frac{\alpha}{h}}|x^s|\cos\theta_y^s & \sqrt{\frac{\alpha}{h}}|x^s|\sin\theta_y^s & \cdots \end{bmatrix},$$

in polar form, where $\theta_x^s = \pi/4$, and $\theta_y^s = \arctan\frac{g-\omega}{g+\omega}$.

If we add inhibitory population self-feedback $-f$ to either model, this additional term appears subtracted from $\alpha$ or $g$ in the real part of the complex eigenvalues, and added to them in all other expressions[2]. Further extensions of this line of analysis will consider lateral inhibitory fan out of the inhibitory - excitatory feedback connections. The $-hI$ block of the coupling matrix $T$ becomes a banded matrix. Similarly, the $gI$ and $-fI$ may be banded, or both full excitatory to excitatory $W$ and full excitatory to inhibitory $V$ coupling blocks may be considered. We conjecture that the phase restrictions of the minimal model will be relaxed with these further degrees of freedom available, so that traveling waves may exist.

### 3.3.1 Acknowledgements

Supported by AFOSR-87-0317. It is a pleasure to acknowledge the support of Walter Freeman and invaluable assistance of Morris Hirsch.

## References

[1] B Baird. A bifurcation theory approach to vector field programming for periodic attractors. In *Proc. Int. Joint Conf. on Neural Networks, Wash. D.C.*, page I381, June 18 1989.

[2] B. Baird. Bifurcation and learning in network models of oscillating cortex. In S. Forest, editor, *Proc. Conf. on Emergent Computation, Los Alamos, May 1989*, 1990. to appear-Physica D.

[3] B. Baird. *Bifurcation Theory Approach to the Analysis and Synthesis of Neural Networks for Engineering and Biological Modelling*. Research Notes in Neural Computing. Springer, 1990.

[4] W.J. Freeman. *Mass Action in the Nervous System*. Academic Press, New York, 1975.

[5] C. M. Grey and W. Singer. Stimulus dependent neuronal oscillations in the cat visual cortex area 17. *Neuroscience [Suppl]*, 22:1301P, 1987.

[6] Lewis B. Haberly and James M. Bower. Olfactory cortex: model circuit for study of associative memory? *Trends in Neuroscience*, 12(7):258, 1989.